# Multiplicative Updates for Classification by Mixture Models

**Lawrence K. Saul**[*] **and Daniel D. Lee**[†]
[*]Department of Computer and Information Science
[†]Department of Electrical Engineering
University of Pennsylvania, Philadelphia, PA 19104

## Abstract

We investigate a learning algorithm for the classification of nonnegative data by mixture models. Multiplicative update rules are derived that directly optimize the performance of these models as classifiers. The update rules have a simple closed form and an intuitive appeal. Our algorithm retains the main virtues of the Expectation-Maximization (EM) algorithm—its guarantee of monotonic improvement, and its absence of tuning parameters—with the added advantage of optimizing a discriminative objective function. The algorithm reduces as a special case to the method of generalized iterative scaling for log-linear models. The learning rate of the algorithm is controlled by the sparseness of the training data. We use the method of nonnegative matrix factorization (NMF) to discover sparse distributed representations of the data. This form of feature selection greatly accelerates learning and makes the algorithm practical on large problems. Experiments show that discriminatively trained mixture models lead to much better classification than comparably sized models trained by EM.

## 1 Introduction

Mixture models[11] have been widely applied to problems in classification. In these problems, one must learn a decision rule mapping feature vectors ($\vec{x}$) to class labels ($y$) given labeled examples. Mixture models are typically used to parameterize class-conditional distributions, $\Pr[\vec{x}|y]$, and then to compute posterior probabilities, $\Pr[y|\vec{x}]$, from Bayes rule. Parameter estimation in these models is handled by an Expectation-Maximization (EM) algorithm[3], a learning procedure that monotonically increases the joint log likelihood, $\sum_n \log \Pr[\vec{x}_n, y_n]$, summed over training examples (indexed by $n$). A virtue of this algorithm is that it does not require the setting of learning rates or other tuning parameters.

A weakness of the above approach is that the model parameters are optimized by maximum likelihood estimation, as opposed to a discriminative criterion more closely related to classification error[14]. In this paper, we derive multiplicative update rules for the parameters of mixture models that directly maximize the discriminative objective function, $\sum_n \log \Pr[y_n|\vec{x}_n]$. This objective function measures the conditional log likelihood that the training examples are correctly classified. Our update rules retain the main virtues of the EM algorithm—its guarantee of monotonic improvement, and its absence of tuning parameters—with the added advantage of optimizing a discriminative cost function. They also have a simple closed form and appealing intuition. The proof of convergence combines ideas from the EM algorithm[3] and methods for generalized and improved iterative

scaling[2, 4].

The approach in this paper is limited to the classification of nonnegative data, since from the constraint of nonnegativity emerges an especially simple learning algorithm. This limitation, though, is not too severe. An abundance of interesting data occurs naturally in this form: for example, the pixel intensities of images, the power spectra of speech, and the word-document counts of text. Real-valued data can also be coerced into this form by addition or exponentiation. Thus we believe the algorithm has broad applicability.

## 2   Mixture models as generative models

Mixture models are typically used as generative models to parameterize probability distributions over feature vectors $\vec{x}_n$. Different mixture models are used to model different classes of data. The parameterized distributions take the form:

$$\Pr[\vec{x}|y = i] = \sum_j W_{ij} \Phi_j(\vec{x}), \tag{1}$$

where the rows of the nonnegative weight matrix $W_{ij}$ are constrained to sum to unity, $\sum_j W_{ij} = 1$, and the basis functions $\Phi_j(\vec{x})$ are properly normalized distributions, such that $\int d\vec{x}\, \Phi_j(\vec{x}) = 1$ for all $j$. The model can be interpreted as the latent variable model,

$$\Pr[\vec{x}|y] = \sum_z \Pr[\vec{x}|z] \Pr[z|y], \tag{2}$$

where the discrete latent variable $z$ indicates which mixture component is used to generate the observed variable $\vec{x}$. In this setting, one identifies $\Phi_j(\vec{x}) = \Pr[\vec{x}|z = j]$ and $W_{ij} = \Pr[z = j|y = i]$. The basis functions, usually chosen from the exponential family, define "bumps" of high probability in the feature space. A popular choice is the multivariate Gaussian distribution:

$$\Phi_j(\vec{x}) = \frac{1}{(2\pi|\Sigma|_j|)^{1/2}} \exp\left\{-\frac{1}{2}(\vec{x} - \vec{\mu}_j)^T \Sigma_j^{-1}(\vec{x} - \vec{\mu}_j)\right\}, \tag{3}$$

with means $\vec{\mu}_j$ and covariance matrices $\Sigma_j$. Gaussian distributions are extremely versatile, but not always the most appropriate. For sparse nonnegative data, a more natural choice is the exponential distribution:

$$\Phi_j(\vec{x}) = \prod_\mu \beta_{j\mu} e^{-\beta_{j\mu} x_\mu}, \tag{4}$$

with parameter vectors $\vec{\beta}_j$. Here, the value of $\mu$ indexes the elements of $\vec{\beta}_j$ and $\vec{x}$. The parameters of these basis functions must be estimated from data.

Generative models can be viewed as a prototype method for classification, with the parameters of each mixture component defining a particular basin of attraction in the feature space. Intuitively, patterns are labeled by the most similar prototype, chosen from among all possible classes. Formally, unlabeled examples are classified by computing posterior probabilities from Bayes' rule,

$$\Pr[y|\vec{x}] = \frac{\Pr[\vec{x}|y]\Pr[y]}{\sum_{y'} \Pr[\vec{x}|y']\Pr[y']}, \tag{5}$$

where $\Pr[y]$ denote the prior probabilities of each class. Examples are classified by the label with the highest posterior probability.

An Expectation-Maximization (EM) algorithm can be used to estimate the parameters of mixture models. The EM algorithm optimizes the joint log likelihood,

$$\mathcal{L}_J = \sum_n \log \Pr[\vec{x}_n | y_n] \Pr[y_n],$$ (6)

summed over training examples. If basis functions are not shared across different classes, then the parameter estimation for $\Pr[\vec{x}|y]$ can be done independently for each class label $y$. This has the tremendous advantage of decomposing the original learning problem into several smaller problems. Moreover, for many types of basis functions, the EM updates have a simple closed form and are guaranteed to improve the joint log likelihood at each iteration. These properties account for the widespread use of mixture models as generative models.

## 3 Mixture models as discriminative models

Mixture models can also be viewed as purely discriminative models. In this view, their purpose is simply to provide a particular way of parameterizing the posterior distribution, $\Pr[y|\vec{x}]$. In this paper, we study posterior distributions of the form:

$$\Pr[y = i | \vec{x}] = \frac{\sum_j W_{ij} \Phi_j(\vec{x})}{\sum_{k\ell} W_{k\ell} \Phi_\ell(\vec{x})}.$$ (7)

The right hand side of this equation defines a valid posterior distribution provided that the mixture weights $W_{ij}$ and basis functions $\Phi_j(\vec{x})$ are nonnegative. Note that for this interpretation, the mixture weights and basis functions do not need to satisfy the more stringent normalization constraints of generative models. We will deliberately exploit this freedom, an idea that distinguishes our approach from previous work on discriminatively trained mixture models[6] and hidden Markov models[5, 12]. In particular, the unnormalized basis functions we use are able to parameterize "saddles" and "valleys" in the feature space, as well as the "bumps" of normalized basis functions. This makes them more expressive than their generative counterparts: examples can not only be attracted to prototypes, but also repelled by opposites.

The posterior distributions in eq. (7) must be further specified by parameterizing the basis functions $\Phi_j(\vec{x})$ as a function of $\vec{x}$. We study basis functions of the form

$$\Phi_j(\vec{x}) = e^{\vec{\theta}_j \cdot \vec{X}},$$ (8)

where $\vec{\theta}_j$ denotes a real-valued vector and $\vec{X}$ denotes a nonnegative and possibly "expanded" representation[14] of the original feature vector. The exponential form in eq. (8) allows us to recover certain generative models as a special case. For example, consider the multivariate Gaussian distribution in eq. (3). By defining the "quadratically expanded" feature vector:

$$\vec{X} = [1, x_1, x_2, \ldots, x_d, x_1 x_1, x_1 x_2, \ldots, x_{d-1} x_d],$$ (9)

we can equate the basis functions in eqs. (3) and (8) by choosing the parameter vectors $\vec{\theta}_j$ to act on $\vec{X}$ in the same way that the means $\vec{\mu}_j$ and covariance matrices $\Sigma_j$ act on $\vec{x}$. The exponential distributions in eq. (4) can be recovered in a similar way. Such generative models provide a cheap way to initialize discriminative models for further training.

## 4 Learning algorithm

Our learning algorithm directly optimizes the performance of the models in eq. (7) as classifiers. The objective function we use for discriminative training is the conditional log likelihood,

$$\mathcal{L}_C = \sum_n \log \Pr[y_n | \vec{x}_n],$$ (10)

summed over training examples. Let $Y_{ni}$ denote the binary matrix whose $ni$th element denotes whether the $n$th training example belongs to the $i$th class. Then we can write the objective function as the difference of two terms, $\mathcal{L}_C = \mathcal{L}_+ - \mathcal{L}_-$, where:

$$\mathcal{L}_+ \quad = \quad \sum_n \log \sum_{ij} Y_{ni} W_{ij} e^{\vec{\theta}_j \cdot \vec{X}_n} \tag{11}$$

$$\mathcal{L}_- \quad = \quad \sum_n \log \sum_{ij} W_{ij} e^{\vec{\theta}_j \cdot \vec{X}_n}. \tag{12}$$

The competition between these terms give rise to a scenario of contrastive learning. It is the subtracted term, $\mathcal{L}_-$, which distinguishes the conditional log likelihood optimized by discriminative training from the joint log likelihood optimized by EM.

Our learning algorithm works by alternately updating the mixture weights and the basis function parameters. Here we simply present the update rules for these parameters; a derivation and proof of convergence are given in the appendix. It is easiest to write the basis function updates in terms of the nonnegative parameters $e^{\theta_{j\mu}}$. The updates then take the simple multiplicative form:

$$W_{ij} \quad \leftarrow \quad W_{ij} \left\{ \left( \frac{\partial \mathcal{L}_+}{\partial W_{ij}} \right) \bigg/ \left( \frac{\partial \mathcal{L}_-}{\partial W_{ij}} \right) \right\}, \tag{13}$$

$$e^{\theta_{j\mu}} \quad \leftarrow \quad e^{\theta_{j\mu}} \left\{ \left( \frac{\partial \mathcal{L}_+}{\partial \theta_{j\mu}} \right) \bigg/ \left( \frac{\partial \mathcal{L}_-}{\partial \theta_{j\mu}} \right) \right\}^{\frac{1}{\eta}} \quad \text{where } \eta = \max_n \sum_\mu X_{n\mu}. \tag{14}$$

It is straightforward to compute the gradients in these ratios and show that they are always nonnegative. (This is a consequence of the nonnegativity constraint on the feature vectors: $X_{n\mu} \geq 0$ for all examples $n$ and feature components $\mu$.) Thus, the nonnegativity constraints on the mixture weights and basis functions are enforced by these multiplicative updates.

The updates have a simple intuition[9] based on balancing opposing terms in the gradient of the conditional log likelihood. In particular, note that the fixed points of this update rule occur at stationary points of the conditional log likelihood—that is, where $\partial \mathcal{L}_C / \partial W_{ij} = 0$ and $\partial \mathcal{L}_C / \partial \theta_{j\mu} = 0$, or equivalently, where $\partial \mathcal{L}_+ / \partial W_{ij} = \partial \mathcal{L}_- / \partial W_{ij}$ and $\partial \mathcal{L}_+ / \partial \theta_{j\mu} = \partial \mathcal{L}_- / \partial \theta_{j\mu}$. The learning rate is controlled by the ratios of these gradients and—additionally, for the basis function updates—by the exponent $1/\eta$, which measures the sparseness of the training data. The value of $\eta$ is the maximum sum of features that occurs in the training data. Thus, sparse feature vectors leads to faster learning, a crucial point to which we will return shortly.

It is worth comparing these multiplicative updates to others in the literature. Jebara and Pentland[6] derived similar updates for mixture weights, but without emphasizing the special form of eq. (13). Others have investigated multiplicative updates by the method of exponentiated gradients (EG)[7]. Our updates do not have the same form as EG updates: in particular, note that the gradients in eqs. (13–14) are not exponentiated. If we use one basis function per class and an identity matrix for the mixture weights, then the updates reduce to the method of generalized iterative scaling[2] for logistic or multinomial regression (also known as maximum entropy modeling). More generally, though, our multiplicative updates can be used to train much more powerful classifiers based on mixture models.

## 5   Feature selection

As previously mentioned, the learning rate for the basis function parameters is controlled by the sparseness of the training data. If this data is not intrinsically sparse, then the multiplicative upates in eqs. (13–14) can be impractically slow (just as the method of iterative

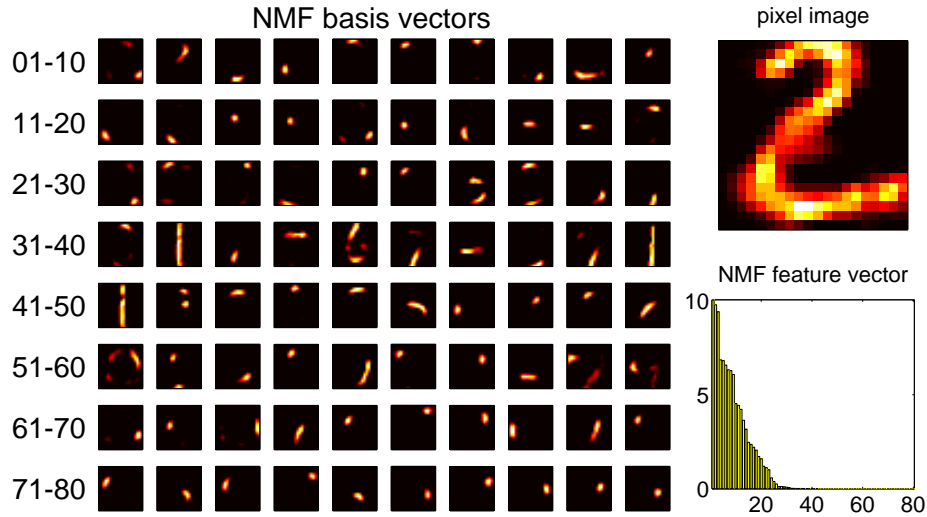

Figure 1: Left: nonnegative basis vectors for handwritten digits discovered by NMF. Right: sparse feature vector for a handwritten '2'. The basis vectors are ordered by their contribution to this image.

scaling). In this case, it is important to discover sparse distributed representations of the data that encode the same information. On large problems, such representations can accelerate learning by several orders of magnitude.

The search for sparse distributed representations can be viewed as a form of feature selection. We have observed that suitably sparse representations can be discovered by the method of nonnegative matrix factorization (NMF)[8]. Let the raw nonnegative (and possibly nonsparse) data be represented by the $D \times N$ matrix $R$, where $D$ is its raw dimensionality and $N$ is the number of training examples. Algorithms for NMF yield a factorization $R \approx BX$, where $B$ is a $D \times d$ nonnegative marix and $X$ is a $d \times N$ nonnegative matrix. In this factorization, the columns of $B$ are interpreted as basis vectors, and the columns of $X$ as coefficients (or new feature vectors). These coefficients are typically very sparse, because the nonnegative basis vectors can only be added in a constructive way to approximate the original data.

The effectiveness of NMF is best illustrated by example. We used the method to discover sparse distributed representations of the MNIST data set of handwritten digits[10]. The data set has 60000 training and 10000 test examples that were deslanted and cropped to form $20 \times 20$ grayscale pixel images. The raw training data was therefore represented by a $D \times N$ matrix, with $D = 400$ and $N = 60000$. The left plot of Fig. 1 shows the $d = 80$ basis vectors discovered by NMF, each plotted as a $20 \times 20$ image. Most of these basis vectors resemble strokes, only a fraction of which are needed to reconstruct any particular image in the training set. For example, only about twenty basis vectors make an appreciable contribution to the handwritten "2" shown in the right plot of Fig. 1. The method of NMF thus succeeds in discovering a highly sparse representation of the original images.

## 6   Results

Models were evaluated on the problem of recognizing handwritten digits from the MNIST data set. From the grayscale pixel images, we generated two sets of feature vectors: one by NMF, with nonnegative features and dimensionality $d_+ = 80$; the other, by principal

components analysis (PCA), with real-valued features and dimensionality $d_{\pm} = 40$. These reduced dimensionality feature vectors were used for both training and testing.

Baseline mixture models for classification were trained by EM algorithms. Gaussian mixture models with diagonal covariance marices were trained on the PCA features, while exponential mixture models (as in eq. (4)) were trained on the NMF features. The mixture models were trained for up to 64 iterations of EM, which was sufficient to ensure a high degree of convergence. Seven baseline classifiers were trained on each feature set, with different numbers of mixture components per digit ($M = 1, 2, 4, 8, 16, 32, 64$). The error rates of these models, indicated by EM-PCA40 and EM-NMF80, are shown in Table 1. Half as many PCA features were used as NMF features so as to equalize the number of fitted parameters in different basis functions.

Mixture models on the NMF features were also trained discriminatively by the multiplicative updates in eqs. (13–14). Models with varying numbers of mixture components per digit ($M = 1, 2, 4, 8, 16$) were trained by 1000 iterations of these updates. Again, this was sufficient to ensure a high degree of convergence; there was no effort at early stopping. The models were initialized by setting $W_{ij} = 1$ and $\vec{\theta}_j = \vec{X}_n$ for randomly selected feature vectors. The results of these experiments, indicated by DT-NMF80, are also shown in Table 1. The results show that the discriminatively trained models classify much better than comparably sized models trained by EM. The ability to learn more compact classifiers appears to be the major advantage of discriminative training. A slight disadvantage is that the resulting classifiers are more susceptible to overtraining.

| model | EM-PCA40 | | EM-NMF80 | | DT-NMF80 | |
|---|---|---|---|---|---|---|
| $M$ | $\mathcal{E}_t$ | $\mathcal{E}_g$ | $\mathcal{E}_t$ | $\mathcal{E}_g$ | $\mathcal{E}_t$ | $\mathcal{E}_g$ |
| 1 | 10.2 | 10.1 | 15.7 | 14.7 | 5.5 | 5.8 |
| 2 | 8.5 | 8.3 | 12.3 | 10.7 | 4.0 | 4.4 |
| 4 | 6.8 | 6.4 | 9.3 | 8.2 | 2.8 | 3.5 |
| 8 | 5.3 | 5.1 | 7.8 | 7.0 | 1.7 | 3.2 |
| 16 | 4.0 | 4.4 | 6.2 | 5.7 | 1.0 | 3.4 |
| 32 | 3.1 | 3.6 | 5.0 | 5.1 | | |
| 64 | 1.9 | 3.1 | 3.9 | 4.2 | | |

Table 1: Classification error rates (%) on the training set ($\mathcal{E}_t$) and the test set ($\mathcal{E}_g$) for mixture models with different numbers of mixture components per digit ($M$). Models in the same row have roughly the same number of fitted parameters.

It is instructive to compare our results to other benchmarks on this data set[10]. Without making use of prior knowledge, better error rates on the test set have been obtained by support vector machines ($\mathcal{E}_g = 1.1\%$), k-nearest neighbor ($\mathcal{E}_g = 2.4\%$), and fully connected multilayer neural networks ($\mathcal{E}_g = 1.6\%$). These results, however, either required storing large numbers of training examples or training significantly larger models. For example, the nearest neighbor and support vector classifiers required storing tens of thousands of training examples (or support vectors), while the neural network had over 120,000 weights. By contrast, the $M = 8$ discriminatively trained mixture model (with $\mathcal{E}_g = 3.2\%$) has less than 6500 iteratively adjusted parameters, and most of its memory footprint is devoted to preprocessing by NMF.

We conclude by describing the problems best suited to the mixture models in this paper. These are problems with many classes, large amounts of data, and little prior knowledge of symmetries or invariances. Support vector machines and nearest neighbor algorithms do not scale well to this regime, and it remains tedious to train large neural networks with unspecified learning rates. By contrast, the compactness of our models and the simplicity of their learning algorithm make them especially attractive.

# A    Proof of convergence

In this appendix, we show that the multiplicative updates from section 4 lead to monotonic improvement in the conditional log likelihood. This guarantee of convergence (to a stationary point) is proved by computing a lower bound on the conditional log likelihood for updated estimates of the mixture weights and basis function parameters. We indicate these updated estimates by $W'_{ij}$ and $\vec{\theta}'_j$, and we indicate the resulting values of the conditional log likelihood and its component terms by $\mathcal{L}'_c$, $\mathcal{L}'_+$, and $\mathcal{L}'_-$. The proof of convergence rests on three simple inequalities applied to $\mathcal{L}'_c$.

The first term in the conditional log likelihood can be lower bounded by Jensen's inequality. The same bound is used here as in the derivation of the EM algorithm[3, 13] for maximum likelihood estimation:

$$\mathcal{L}'_+ = \sum_n \log \sum_{ij} Y_{ni} W'_{ij} e^{\vec{\theta}'_j \cdot \vec{X}_n} \geq \sum_{nij} P^+_{nij} \log \left[ \frac{Y_{ni} W'_{ij} e^{\vec{\theta}'_j \cdot \vec{X}_n}}{P^+_{nij}} \right]. \tag{15}$$

The right hand side of this inequality introduces an auxiliary probability distribution $P^+_{nij}$ for each example in the training set. The bound holds for arbitrary distributions, provided they are properly normalized: $\sum_{ij} P^+_{nij} = 1$ for all $n$.

The second term in the conditional log likelihood occurs with a minus sign, so for this term we require an upper bound. The same bounds can be used here as in derivations of iterative scaling[1, 2, 4, 13]. Note that the logarithm function is upper bounded by: $\log z \leq z - 1$ for all $z > 0$. We can therefore write:

$$\mathcal{L}'_- - \mathcal{L}_- = \sum_n \log \left( \frac{\sum_{ij} W'_{ij} e^{\vec{\theta}'_j \cdot \vec{X}_n}}{\sum_{k\ell} W_{k\ell} e^{\vec{\theta}_\ell \cdot \vec{X}_n}} \right) \leq \sum_n \left[ \frac{\sum_{ij} W'_{ij} e^{\vec{\theta}'_j \cdot \vec{X}_n}}{\sum_{k\ell} W_{k\ell} e^{\vec{\theta}_\ell \cdot \vec{X}_n}} - 1 \right]. \tag{16}$$

To further bound the right hand side of eq. (16), we make the following observation: though the exponentials $e^{\vec{\theta}_j \cdot \vec{X}_n}$ are *convex* functions of the parameter vector $\vec{\theta}'_j$ with elements $\theta'_{j\mu}$, they are *concave* functions of the "warped" parameter vector $e^{\eta \vec{\theta}'_j}$ with elements $e^{\eta \theta'_{j\mu}}$, where $\eta$ is defined by eq. (14). (The validity of this observation hinges on the nonnegativity of the feature vectors $\vec{X}_n$.) It follows that for any example in the training set, the exponential $e^{\vec{\theta}'_j \cdot \vec{X}_n}$ is upper bounded by its linearized expansion around $e^{\eta \theta'_{j\mu}} = e^{\eta \theta_{j\mu}}$, given by:

$$e^{\vec{\theta}'_j \cdot \vec{X}_n} \leq e^{\vec{\theta}_j \cdot \vec{X}_n} + \sum_\mu \left( e^{\eta \theta'_{j\mu}} - e^{\eta \theta_{j\mu}} \right) \left( \frac{X_{n\mu} e^{\vec{\theta}_j \cdot \vec{X}_n}}{\eta e^{\eta \theta_{j\mu}}} \right). \tag{17}$$

The last term in parentheses in eq. (17) is the derivative of $e^{\vec{\theta}_j \cdot \vec{X}_n}$ with respect to the independent variable $e^{\eta \theta_{j\mu}}$, computed by the chain rule. Tighter bounds are possible than eq. (17), but at the expense of more complicated update rules.

Combining the above inequalities with a judicious choice for the auxiliary parameters $P^+_{nij}$, we obtain a proof of convergence for the multiplicative updates in eqs. (13–14). Let:

$$P^+_{nij} = \left( \sum_\ell W_{i\ell} e^{\vec{\theta}_\ell \cdot \vec{X}_n} \right)^{-1} Y_{ni} W_{ij} e^{\vec{\theta}_j \cdot \vec{X}_n}, \tag{18}$$

$$P^-_{nij} = \left( \sum_{k\ell} W_{k\ell} e^{\vec{\theta}_\ell \cdot \vec{X}_n} \right)^{-1} W_{ij} e^{\vec{\theta}_j \cdot \vec{X}_n}. \tag{19}$$

Eq. (18) sets the auxiliary parameters $P^+_{nij}$, while eq. (19) defines an analogous distribution $P^-_{nij}$ for the opposing term in the conditional log likelihood. (This will prove to be a useful notation.) Combining these definitions with eqs. (15–17) and rearranging terms, we obtain the following inequality:

$$\mathcal{L}'_c - \mathcal{L}_c \geq \sum_{nij} P^+_{nij} \left\{ \log \left[ \frac{W'_{ij}}{W_{ij}} \right] + \left( \vec{\theta}'_j - \vec{\theta}_j \right) \cdot \vec{X}_n \right\}$$

$$- \sum_{nij} P^-_{nij} \left\{ \frac{W'_{ij}}{W_{ij}} - 1 + \frac{W'_{ij}}{W_{ij}} \sum_\mu X_{n\mu} \left( \frac{e^{\eta(\theta'_{j\mu} - \theta_{j\mu})} - 1}{\eta} \right) \right\} \quad (20)$$

Both sides of the inequality vanish (yielding an equality) if $W'_{ij} = W_{ij}$ and $\vec{\theta}'_j = \vec{\theta}_j$. We derive the update rules by maximizing the right hand side of this inequality. Maximizing the right hand side with respect to $W'_{ij}$ while holding the basis function parameters fixed yields the update, eq. (13). Likewise, maximizing the right hand side with respect to $\vec{\theta}'_j$ while holding the mixture weights fixed yields the update, eq. (14). Since these choices for $W'_{ij}$ and $\vec{\theta}'$ lead to positive values on the right hand side of the inequality (except at fixed points), it follows that the multiplicative updates in eqs. (13–14) lead to monotonic improvement in the conditional log likelihood.

# References

[1] M. Collins, R. Schapire, and Y. Singer (2000). Logistic regression, adaBoost, and Bregman distances. In *Proceedings of the Thirteenth Annual Conference on Computational Learning Theory*.

[2] J. N. Darroch and D. Ratcliff (1972). Generalized iterative scaling for log-linear models. *Annals of Mathematical Statistics* **43**:1470–1480.

[3] A. P. Dempster, N. M. Laird, and D. B. Rubin (1977). Maximum likelihood from incomplete data via the EM algorithm. *J. Royal Stat. Soc. B* **39**: 1–37.

[4] S. Della Pietra, V. Della Pietra, and J. Lafferty (1997). Inducing features of random fields. *IEEE Transactions on Pattern Analysis and Machine Intelligence* 19(4): 380–393.

[5] P.S. Gopalakrishnan, D. Kanevsky, A. Ndas and D. Nahamoo (1991). An inequality for rational functions with applications to some statistical estimation problems. IEEE Transactions on Information Theory **37**: 107–113.

[6] T. Jebara and A. Pentland (1998). Maximum conditional likelihood via bound maximization and the CEM algorithm. In M. Kearns, S. Solla, and D. Cohn (eds.). *Advances in Neural Information Processing Systems 11*, 494–500. MIT Press: Cambridge, MA.

[7] J. Kivinen and M. Warmuth (1997). Additive versus exponentiated gradient updates for linear prediction. *Journal of Information and Computation* **132**: 1–64.

[8] D. D. Lee and H. S. Seung (1999). Learning the parts of objects with nonnegative matrix factorization. *Nature* **401**: 788–791.

[9] D. D. Lee and H. S. Seung (2000). Algorithms for nonnegative matrix factorization. In T. Leen, T. Dietterich, and V. Tresp (eds.). *Advances in Neural Information Processing Systems 13*. MIT Press: Cambridge, MA.

[10] Y.LeCun, L. Jackel, L.Bottou, A.Brunot, C.Cortes, J. Denker, H.Drucker, I.Guyon, U. Muller, E.Sackinger, P.Simard, and V.Vapnik (1995). A comparison of learning algorithms for handwritten digit recognition. In F.Fogelman and P.Gallinari (eds.). *International Conference on Artificial Neural Networks*, 1995, Paris: 53–60.

[11] G. McLachlan and K. Basford (1988). *Mixture Models: Inference and Applications to Clustering.* Marcel Dekker.

[12] Y. Normandin (1991). *Hidden Markov Models, Maximum Mutual Information Estimation and the Speech Recognition Problem.* Ph.D. Thesis, McGill University, Montreal.

[13] J. A. O'Sullivan (1998). Alternating minimization algorithms: from Blahut-Arimoto to Expectation-Maximization. In A. Vardy (ed.). *Codes, Curves, and Signals: Common Threads in Communications.* Kluwer: Norwell, MA.

[14] V. Vapnik (1999). *The Nature of Statistical Learning Theory.* Springer Verlag.
